# Learning Convolutional Feature Hierarchies for Visual Recognition

**Koray Kavukcuoglu**[1], **Pierre Sermanet**[1], **Y-Lan Boureau**[2,1],
**Karol Gregor**[1], **Michaël Mathieu**[1], **Yann LeCun**[1]
[1] Courant Institute of Mathematical Sciences, New York University
[2] INRIA - Willow project-team*
{koray,sermanet,ylan,kgregor,yann}@cs.nyu.edu, mmathieu@clipper.ens.fr

## Abstract

We propose an unsupervised method for learning multi-stage hierarchies of sparse convolutional features. While sparse coding has become an increasingly popular method for learning visual features, it is most often trained at the patch level. Applying the resulting filters convolutionally results in highly redundant codes because overlapping patches are encoded in isolation. By training convolutionally over large image windows, our method reduces the redudancy between feature vectors at neighboring locations and improves the efficiency of the overall representation. In addition to a linear decoder that reconstructs the image from sparse features, our method trains an efficient feed-forward encoder that predicts quasi-sparse features from the input. While patch-based training rarely produces anything but oriented edge detectors, we show that convolutional training produces highly diverse filters, including center-surround filters, corner detectors, cross detectors, and oriented grating detectors. We show that using these filters in multi-stage convolutional network architecture improves performance on a number of visual recognition and detection tasks.

## 1 Introduction

Over the last few years, a growing amount of research on visual recognition has focused on learning low-level and mid-level features using unsupervised learning, supervised learning, or a combination of the two. The ability to learn multiple levels of good feature representations in a hierarchical structure would enable the automatic construction of sophisticated recognition systems operating, not just on natural images, but on a wide variety of modalities. This would be particularly useful for sensor modalities where our lack of intuition makes it difficult to engineer good feature extractors.

The present paper introduces a new class of techniques for learning features extracted though *convolutional filter banks*. The techniques are applicable to Convolutional Networks and their variants, which use multiple stages of trainable convolutional filter banks, interspersed with non-linear operations, and spatial feature pooling operations [1, 2]. While ConvNets have traditionally been trained in supervised mode, a number of recent systems have proposed to use unsupervised learning to pre-train the filters, followed by supervised fine-tuning. Some authors have used convolutional forms of Restricted Boltzmann Machines (RBM) trained with contrastive divergence [3], but many of them have relied on sparse coding and sparse modeling [4, 5, 6]. In sparse coding, a sparse feature vector $z$ is computed so as to best reconstruct the input $x$ through a linear operation with a *learned dictionary matrix* $\mathcal{D}$. The inference procedure produces a code $z^*$ by minimizing an energy function:

$$\mathcal{L}(x, z, \mathcal{D}) = \frac{1}{2}||x - \mathcal{D}z||_2^2 + |z|_1, \quad z^* = \arg\min_z \mathcal{L}(x, z, \mathcal{D}) \tag{1}$$

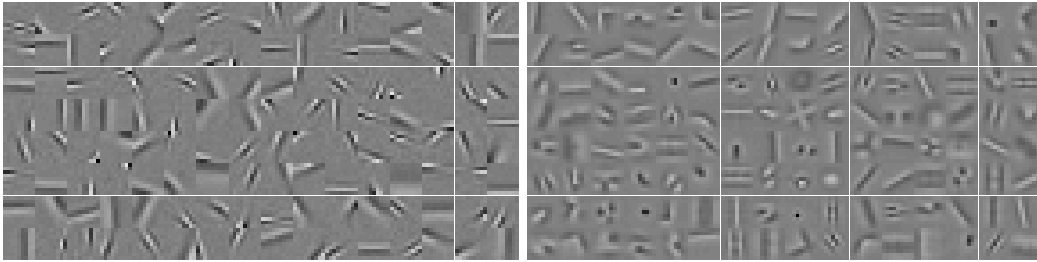

Figure 1: **Left:** A dictionary with 128 elements, learned with patch based sparse coding model. **Right:** A dictionary with 128 elements, learned with convolutional sparse coding model. The dictionary learned with the convolutional model spans the orientation space much more uniformly. In addition it can be seen that the diversity of filters obtained by convolutional sparse model is much richer compared to patch based one.

The dictionary is obtained by minimizing the energy 1 wrt $\mathcal{D}$: $\min_{z,\mathcal{D}} \mathcal{L}(x, z, \mathcal{D})$ averaged over a training set of input samples. There are two problems with the traditional sparse modeling method when training convolutional filter banks: 1: the representations of whole images are highly redundant because the training and the inference are performed at the patch level; 2: the inference for a whole image is computationally expensive.

**First problem.** In most applications of sparse coding to image analysis [7, 8], the system is trained on *single image patches* whose dimensions match those of the filters. After training, patches in the image are processed separately. This procedure completely ignores the fact that the filters are eventually going to be used in a convolutional fashion. Learning will produce a dictionary of filters that are essentially shifted versions of each other over the patch, so as to reconstruct each patch in isolation. Inference is performed on all (overlapping) patches independently, which produces a very highly redundant representation for the whole image. To address this problem, we apply sparse coding to the entire image at once, and we view the dictionary as a convolutional filter bank:

$$\mathcal{L}(x, z, \mathcal{D}) = \frac{1}{2}||x - \sum_{k=1}^{K} \mathcal{D}_k * z_k||_2^2 + |z|_1, \tag{2}$$

where $\mathcal{D}_k$ is an $s \times s$ 2D filter kernel, $x$ is a $w \times h$ image (instead of an $s \times s$ patch), $z_k$ is a 2D feature map of dimension $(w + s - 1) \times (h + s - 1)$, and "$*$" denotes the discrete convolution operator. Convolutional Sparse Coding has been used by several authors, notably [6].

To address the **second problem**, we follow the idea of [4, 5], and use a trainable, feed-forward, non-linear *encoder* module to produce a fast approximation of the sparse code. The new energy function includes a code prediction error term:

$$\mathcal{L}(x, z, \mathcal{D}, W) = \frac{1}{2}||x - \sum_{k=1}^{K} \mathcal{D}_k * z_k||_2^2 + \sum_{k=1}^{K} ||z_k - f(W^k * x)||_2^2 + |z|_1, \tag{3}$$

where $z^* = \arg\min_z \mathcal{L}(x, z, \mathcal{D}, W)$ and $W^k$ is an encoding convolution kernel of size $s \times s$, and $f$ is a point-wise non-linear function. Two crucially important questions are the form of the non-linear function $f$, and the optimization method to find $z^*$. Both questions will be discussed at length below.

The contribution of this paper is to address both issues simultaneously, thus allowing convolutional approaches to sparse coding to scale up, and opening the road to real-time applications.

## 2 Algorithms and Method

In this section, we analyze the benefits of convolutional sparse coding for object recognition systems, and propose convolutional extensions to the coordinate descent sparse coding (CoD) [9] algorithm and the dictionary learning procedure.

### 2.1 Learning Convolutional Dictionaries

The key observation for modeling convolutional filter banks is that the convolution of a signal with a given kernel can be represented as a matrix-vector product by constructing a special Toeplitz-structured matrix for each dictionary element and concatenating all such matrices to form a new

dictionary. Any existing sparse coding algorithm can then be used. Unfortunately, this method incurs a cost, since the size of the dictionary then depends on the size of the input signal. Therefore, it is advantageous to use a formulation based on convolutions rather than following the naive method outlined above. In this work, we use the coordinate descent sparse coding algorithm [9] as a starting point and generalize it using convolution operations. Two important issues arise when learning convolutional dictionaries: 1. The boundary effects due to convolutions need to be properly handled. 2. The derivative of equation 2 should be computed efficiently. Since the loss is not jointly convex in $\mathcal{D}$ and $z$, but is convex in each variable when the other one is kept fixed, sparse dictionaries are usually learned by an approach similar to block coordinate descent, which alternatively minimizes over $z$ and $\mathcal{D}$ (e.g., see [10, 8, 4]). One can use either batch [7] (by accumulating derivatives over many samples) or online updates [8, 6, 5] (updating the dictionary after each sample). In this work, we use a stochastic online procedure for updating the dictionary elements.

The updates to the dictionary elements, calculated from equation 2, are sensitive to the boundary effects introduced by the convolution operator. The code units that are at the boundary might grow much larger compared to the middle elements, since the outermost boundaries of the reconstruction take contributions from only a single code unit, compared to the middle ones that combine $s \times s$ units. Therefore the reconstruction error, and correspondingly the derivatives, grow proportionally larger. One way to properly handle this situation is to apply a mask on the derivatives of the reconstruction error wrt $z$: $D^T * (x - \mathcal{D} * z)$ is replaced by $D^T * (mask(x) - \mathcal{D} * z)$, where $mask$ is a term-by-term multiplier that either puts zeros or gradually scales down the boundaries.

---

**Algorithm 1** Convolutional extension to coordinate descent sparse coding[9]. A subscript index (set) of a matrix represent a particular element. For slicing the $4D$ tensor $S$ we adopt the MATLAB notation for simplicity of notation.

---

   **function ConvCoD**$(x, \mathcal{D}, \alpha)$
      **Set:** $S = \mathcal{D}^T * \mathcal{D}$
      **Initialize:** $z = 0$; $\beta = \mathcal{D}^T * mask(x)$
      **Require:** $h_\alpha$ : smooth thresholding function.
      **repeat**
         $\bar{z} = h_\alpha(\beta)$
         $(k, p, q) = \arg\max_{i,m,n} |z_{imn} - \bar{z}_{imn}|$ ($k$ : dictionary index, $(p.q)$ : location index)
         $bi = \beta_{kpq}$
         $\beta = \beta + (z_{kpq} - \bar{z}_{kpq}) \times align(S(:, k, :, :), (p, q))$
         $z_{kpq} = \bar{z}_{kpq}, \beta_{kpq} = bi$
      **until** change in $z$ is below a threshold
   **end function**

---

The second important point in training convolutional dictionaries is the computation of the $S = D^T * D$ operator. For most algorithms like coordinate descent [9], FISTA [11] and matching pursuit [12], it is advantageous to store the similarity matrix ($S$) explicitly and use a single column at a time for updating the corresponding component of code $z$. For convolutional modeling, the same approach can be followed with some additional care. In patch based sparse coding, each element $(i, j)$ of $S$ equals the dot product of dictionary elements $i$ and $j$. Since the similarity of a pair of dictionary elements has to be also considered in spatial dimensions, each term is expanded as *"full"* convolution of two dictionary elements $(i, j)$, producing $2s - 1 \times 2s - 1$ matrix. It is more convenient to think about the resulting matrix as a $4D$ tensor of size $K \times K \times 2s - 1 \times 2s - 1$. One should note that, depending on the input image size, proper alignment of corresponding column of this tensor has to be applied in the $z$ space. One can also use the steepest descent algorithm for finding the solution to convolutional sparse coding given in equation 2, however using this method would be orders of magnitude slower compared to specialized algorithms like CoD [9] and the solution would never contain exact zeros. In algorithm 1 we explain the extension of the coordinate descent algorithm [9] for convolutional inputs. Having formulated convolutional sparse coding, the overall learning procedure is simple stochastic (online) gradient descent over dictionary $\mathcal{D}$:

$$\forall x^i \in \mathcal{X} \text{ training set}: \quad z^* = \arg\min_z \mathcal{L}(x^i, z, \mathcal{D}), \quad \mathcal{D} \leftarrow \mathcal{D} - \eta \frac{\partial \mathcal{L}(x^i, z^*, \mathcal{D})}{\partial \mathcal{D}} \qquad (4)$$

The columns of $\mathcal{D}$ are normalized after each iteration. A convolutional dictionary with 128 elements which was trained on images from Berkeley dataset [13] is shown in figure 1.

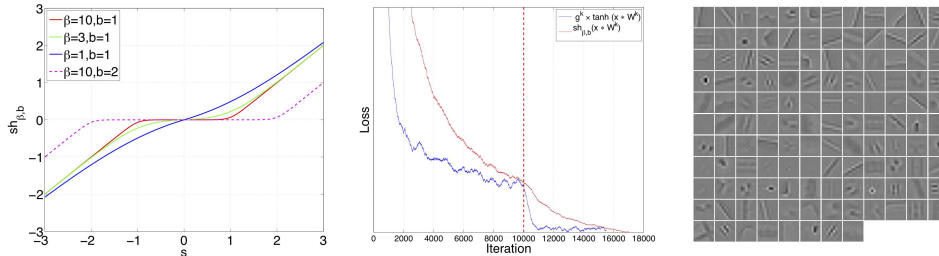

Figure 2: **Left:** Smooth shrinkage function. Parameters $\beta$ and $b$ control the smoothness and location of the kink of the function. As $\beta \to \infty$ it converges more closely to soft thresholding operator. **Center:** Total loss as a function of number of iterations. The vertical dotted line marks the iteration number when diagonal hessian approximation was updated. It is clear that for both encoder functions, hessian update improves the convergence significantly. **Right:** 128 convolutional filters $(W)$ learned in the encoder using smooth shrinkage function. The decoder of this system is shown in image 1.

## 2.2 Learning an Efficient Encoder

In [4], [14] and [15] a feedforward regressor was trained for fast approximate inference. In this work, we extend their encoder module training to convolutional domain and also propose a new encoder function that approximates sparse codes more closely. The encoder used in [14] is a simple feedforward function which can also be seen as a small convolutional neural network: $\tilde{z} = g^k \times tanh(x * W^k)$ $(k = 1..K)$. This function has been shown to produce good features for object recognition [14], however it does not include a shrinkage operator, thus its ability to produce sparse representations is very limited. Therefore, we propose a different encoding function with a shrinkage operator. The standard soft thresholding operator has the nice property of producing exact zeros around the origin, however for a very wide region, the derivatives are also zero. In order to be able to train a filter bank that is applied to the input before the shrinkage operator, we propose to use an encoder with a smooth shrinkage operator $\tilde{z} = sh_{\beta^k,b^k}(x * W^k)$ where $k = 1..K$ and :

$$sh_{\beta^k,b^k}(s) = sign(s) \times 1/\beta^k \log(\exp(\beta^k \times b^k) + \exp(\beta^k \times |s|) - 1) - b^k \qquad (5)$$

Note that each $\beta^k$ and $b^k$ is a singleton per each feature map $k$. The shape of the smooth shrinkage operator is given in figure 2 for several different values of $\beta$ and $b$. It can be seen that $\beta$ controls the smoothness of the kink of shrinkage operator and $b$ controls the location of the kink. The function is guaranteed to pass through the origin and is antisymmetric. The partial derivatives $\frac{\partial sh}{\partial \beta}$ and $\frac{\partial sh}{\partial b}$ can be easily written and these parameters can be learned from data.

Updating the parameters of the encoding function is performed by minimizing equation 3. The additional cost term penalizes the squared distance between optimal code $z$ and prediction $\tilde{z}$. In a sense, training the encoder module is similar to training a ConvNet. To aid faster convergence, we use stochastic diagonal Levenberg-Marquardt method [16] to calculate a positive diagonal approximation to the hessian. We update the hessian approximation every 10000 samples and the effect of hessian updates on the total loss is shown in figure 2. It can be seen that especially for the $tanh$ encoder function, the effect of using second order information on the convergence is significant.

## 2.3 Patch Based vs Convolutional Sparse Modeling

Natural images, sounds, and more generally, signals that display translation invariance in any dimension, are better represented using convolutional dictionaries. The convolution operator enables the system to model local structures that appear anywhere in the signal. For example, if $k \times k$ image patches are sampled from a set of natural images, an edge at a given orientation may appear at any location, forcing local models to allocate multiple dictionary elements to represent a single underlying orientation. By contrast, a convolutional model only needs to record the oriented structure once, since dictionary elements can be used at all locations. Figure 1 shows atoms from patch-based and convolutional dictionaries comprising the same number of elements. The convolutional dictionary does not waste resources modeling similar filter structure at multiple locations. Instead, it models more orientations, frequencies, and different structures including center-surround filters, double center-surround filters, and corner structures at various angles.

In this work, we present two encoder architectures, 1. steepest descent sparse coding with $tanh$ encoding function using $g^k \times tanh(x * W^k)$, 2. convolutional CoD sparse coding with $shrink$

encoding function using $sh_{\beta,b}(x * W^k)$. The time required for training the first system is much higher than for the second system due to steepest descent sparse coding. However, the performance of the encoding functions are almost identical.

## 2.4 Multi-stage architecture

Our convolutional encoder can be used to replace patch-based sparse coding modules used in multi-stage object recognition architectures such as the one proposed in our previous work [14]. Building on our previous findings, for each stage, the encoder is followed by and absolute value rectification, contrast normalization and average subsampling. **Absolute Value Rectification** is a simple pointwise absolute value function applied on the output of the encoder. **Contrast Normalization** is the same operation used for pre-processing the images. This type of operation has been shown to reduce the dependencies between components [17, 18] (feature maps in our case). When used in between layers, the mean and standard deviation is calculated across all feature maps with a $9 \times 9$ neighborhood in spatial dimensions. The last operation, **average pooling** is simply a spatial pooling operation that is applied on each feature map independently.

One or more additional stages can be stacked on top of the first one. Each stage then takes the output of its preceding stage as input and processes it using the same series of operations with different architectural parameters like size and connections. When the input to a stage is a series of feature maps, each output feature map is formed by the summation of multiple filters.

In the next sections, we present experiments showing that using convolutionally trained encoders in this architecture lead to better object recognition performance.

## 3 Experiments

We closely follow the architecture proposed in [14] for object recognition experiments. As stated above, in our experiments, we use two different systems: **1.** Steepest descent sparse coding with $tanh$ encoder: $\mathbf{SD}^{tanh}$. **2.** Coordinate descent sparse coding with $shrink$ encoder: $\mathbf{CD}^{shrink}$. In the following, we give details of the unsupervised training and supervised recognition experiments.

### 3.1 Object Recognition using Caltech 101 Dataset

The Caltech-101 dataset [19] contains up to 30 training images per class and each image contains a single object. We process the images in the dataset as follows: **1.** Each image is converted to gray-scale and resized so that the largest edge is $151$. **2.** Images are contrast normalized to obtain locally zero mean and unit standard deviation input using a $9 \times 9$ neighborhood. **3.** The short side of each image is zero padded to $143$ pixels. We report the results in Table 1 and 2. All results in these tables are obtained using 30 training samples per class and 5 different choices of the training set. We use the background class during training and testing.

**Architecture :** We use the unsupervised trained encoders in a multi-stage system identical to the one proposed in [14]. At first layer 64 features are extracted from the input image, followed by a second layers that produces 256 features. Second layer features are connected to fist layer features through a sparse connection table to break the symmetry and to decrease the number of parameters.

**Unsupervised Training :** The input to unsupervised training consists of contrast normalized gray-scale images [20] obtained from the Berkeley segmentation dataset [13]. Contrast normalization consists of processing each feature map value by removing the mean and dividing by the standard deviation calculated around $9 \times 9$ region centered at that value over all feature maps.

**First Layer:** We have trained both systems using $64$ dictionary elements. Each dictionary item is a $9 \times 9$ convolution kernel. The resulting system to be solved is a $64$ times overcomplete sparse coding problem. Both systems are trained for 10 different sparsity values ranging between $0.1$ and $3.0$.

**Second Layer:** Using the $64$ feature maps output from the first layer encoder on Berkeley images, we train a second layer convolutional sparse coding. At the second layer, the number of feature maps is 256 and each feature map is connected to $16$ randomly selected input features out of $64$. Thus, we aim to learn $4096$ convolutional kernels at the second layer. To the best of our knowledge, none of the previous convolutional RBM [3] and sparse coding [6] methods have learned such a large number of dictionary elements. Our aim is motivated by the fact that using such large number of elements and using a linear classifier [14] reports recognition results similar to [3] and [6]. In both of these studies a more powerful Pyramid Match Kernel SVM classifier [21] is used to match the same level of performance. Figure 3 shows 128 filters that connect to 8 first layer features. Each

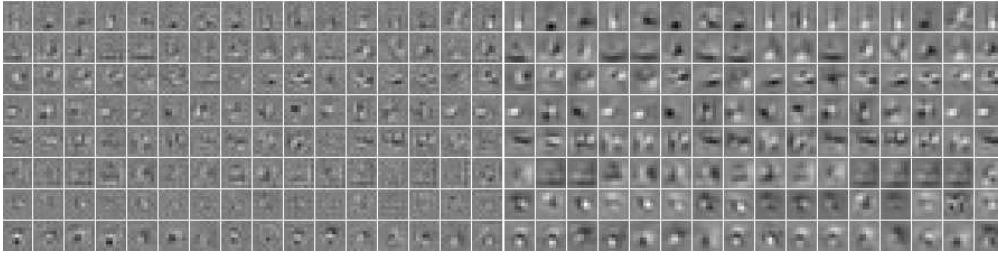

Figure 3: Second stage filters. **Left:** Encoder kernels that correspond to the dictionary elements. **Right:** 128 dictionary elements, each row shows 16 dictionary elements, connecting to a single second layer feature map. It can be seen that each group extracts similar type of features from their corresponding inputs.

row of filters connect a particular second layer feature map. It is seen that each row of filters extract similar features since their output response is summed together to form one output feature map.

| | **Logistic Regression Classifier** | | |
|---|---|---|---|
| | $\mathbf{SD}^{tanh}$ | $\mathbf{CD}^{shrink}$ | **PSD [14]** |
| **U** | $57.1 \pm 0.6\%$ | $57.3 \pm 0.5\%$ | $52.2\%$ |
| $\mathbf{U}^{+}$ | $57.6 \pm 0.4\%$ | $56.4 \pm 0.5\%$ | $54.2\%$ |

Table 1: Comparing $\mathbf{SD}^{tanh}$ encoder to $\mathbf{CD}^{shrink}$ encoder on Caltech 101 dataset using a single stage architecture. Each system is trained using 64 convolutional filters. The recognition accuracy results shown are very similar for both systems.

**One Stage System:** We train 64 convolutional unsupervised features using both $\mathbf{SD}^{tanh}$ and $\mathbf{CD}^{shrink}$ methods. We use the encoder function obtained from this training followed by absolute value rectification, contrast normalization and average pooling. The convolutional filters used are $9 \times 9$. The average pooling is applied over a $10 \times 10$ area with 5 pixel stride. The output of first layer is then $64 \times 26 \times 26$ and fed into a logistic regression classifier and Lazebnik's PMK-SVM classifier [21] (that is, the spatial pyramid pipeline is used, using our features to replace the SIFT features).

**Two Stage System:** We train 4096 convolutional filters with $\mathbf{SD}^{tanh}$ method using 64 input feature maps from first stage to produce 256 feature maps. The second layer features are also $9 \times 9$, producing $256 \times 18 \times 18$ features. After applying absolute value rectification, contrast normalization and average pooling (on a $6 \times 6$ area with stride 4), the output features are $256 \times 4 \times 4$ (4096) dimensional. We only use multinomial logistic regression classifier after the second layer feature extraction stage.

We denote unsupervised trained one stage systems with $U$, two stage unsupervised trained systems with $UU$ and "$+$" represents supervised training is performed afterwards. $R$ stands for randomly initialized systems with no unsupervised training.

| **Logistic Regression Classifier** | |
|---|---|
| **PSD [14]** ($\mathbf{UU}$) | $63.7$ |
| **PSD [14]** ($\mathbf{U}^{+}\mathbf{U}^{+}$) | $65.5$ |
| $\mathbf{SD}^{tanh}$ ($\mathbf{UU}$) | $65.3 \pm 0.9\%$ |
| $\mathbf{SD}^{tanh}$ ($\mathbf{U}^{+}\mathbf{U}^{+}$) | $66.3 \pm 1.5\%$ |

| **PMK-SVM [21] Classifier:** Hard quantization + multiscale pooling + intersection kernel SVM | |
|---|---|
| **SIFT [21]** | $64.6 \pm 0.7\%$ |
| **RBM [3]** | $66.4 \pm 0.5\%$ |
| **DN [6]** | $66.9 \pm 1.1\%$ |
| $\mathbf{SD}^{tanh}$ ($\mathbf{U}$) | $65.7 \pm 0.7\%$ |

Table 2: Recognition accuracy on Caltech 101 dataset using a variety of different feature representations using two stage systems and two different classifiers.

Comparing our $U$ system using both $\mathbf{SD}^{tanh}$ and $\mathbf{CD}^{shrink}$ ($57.1\%$ and $57.3\%$) with the $52.2\%$ reported in [14], we see that convolutional training results in significant improvement. With two layers of purely unsupervised features ($UU$, $65.3\%$), we even achieve the same performance as the patch-based model of Jarrett et al. [14] after supervised fine-tuning ($63.7\%$). Moreover, with additional supervised fine-tuning ($U^{+}U^{+}$) we match or perform very close to ($66.3\%$) similar models [3, 6]

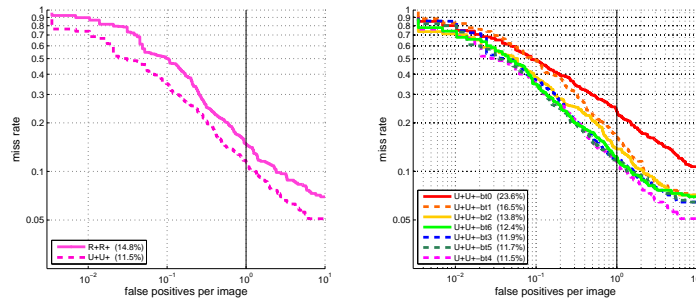

Figure 4: Results on the INRIA dataset with per-image metric. **Left:** Comparing two best systems with unsupervised initialization ($UU$) vs random initialization ($RR$). **Right:** Effect of bootstrapping on final performance for unsupervised initialized system.

with two layers of convolutional feature extraction, even though these models use the more complex spatial pyramid classifier (PMK-SVM) instead of the logistic regression we have used; the spatial pyramid framework comprises a codeword extraction step and an SVM, thus effectively adding one layer to the system. We get $65.7\%$ with a spatial pyramid on top of our single-layer $U$ system (with 256 codewords jointly encoding $2 \times 2$ neighborhoods of our features by hard quantization, then max pooling in each cell of the pyramid, with a linear SVM, as proposed by authors in [22]).

Our experiments have shown that sparse features achieve superior recognition performance compared to features obtained using a dictionary trained by a patch-based procedure as shown in Table 2. It is interesting to note that the improvement is larger when using feature extractors trained in a purely unsupervised way, than when unsupervised training is followed by a supervised training phase ($57.1$ to $57.6$). Recalling that the supervised tuning is a *convolutional* procedure, this last training step might have the additional benefit of decreasing the redundancy between patch-based dictionary elements. On the other hand, this contribution would be minor for dictionaries which have already been trained convolutionally in the unsupervised stage.

### 3.2 Pedestrian Detection

We train and evaluate our architecture on the INRIA Pedestrian dataset [23] which contains 2416 positive examples (after mirroring) and 1218 negative full images. For training, we also augment the positive set with small translations and scale variations to learn invariance to small transformations, yielding 11370 and 1000 positive examples for training and validation respectively. The negative set is obtained by sampling patches from negative full images at random scales and locations. Additionally, we include samples from the positive set with larger and smaller scales to avoid false positives from very different scales. With these additions, the negative set is composed of 9001 training and 1000 validation samples.

### Architecture and Training

A similar architecture as in the previous section was used, with 32 filters, each $7 \times 7$ for the first layer and 64 filters, also $7 \times 7$ for the second layer. We used $2 \times 2$ average pooling between each layer. A fully connected linear layer with 2 output scores (for pedestrian and background) was used as the classifier. We trained this system on $78 \times 38$ inputs where pedestrians are approximately 60 pixels high. We have trained our system with and without unsupervised initialization, followed by fine-tuning of the entire architecture in supervised manner. Figure 5 shows comparisons of our system with other methods as well as the effect of unsupervised initialization.

After one pass of unsupervised and/or supervised training, several bootstrapping passes were performed to augment the negative set with the 10 most offending samples on each full negative image and the bigger/smaller scaled positives. We select the most offending sample that has the biggest opposite score. We limit the number of extracted false positives to 3000 per bootstrapping pass. As [24] showed, the number of bootstrapping passes matters more than the initial training set. We find that the best results were obtained after four passes, as shown in figure 5 improving from $23.6\%$ to $11.5\%$.

### Per-Image Evaluation

Performance on the INRIA set is usually reported with the per-window methodology to avoid post-processing biases, assuming that better per-window performance yields better per-image perfor-

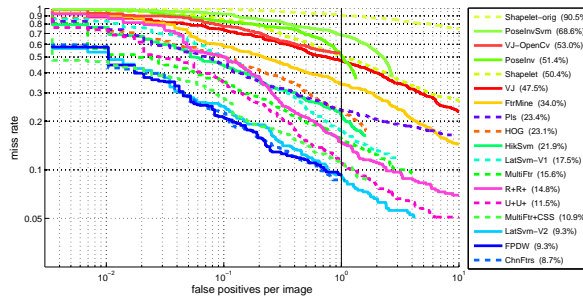

Figure 5: Results on the INRIA dataset with per-image metric. These curves are computed from the bounding boxes and confidences made available by [25]. Comparing our two best systems labeled $(U^+U^+$ and $R^+R^+)$with all the other methods.

mance. However [25] empirically showed that the per-window methodology fails to predict the performance per-image and therefore is not adequate for real applications. Thus, we evaluate the per-image accuracy using the source code available from [25], which matches bounding boxes with the 50% PASCAL matching measure ($\frac{intersection}{union} > 0.5$).

In figure 5, we compare our best results (11.5%) to the latest state-of-the-art results (8.7%) gathered and published on the Caltech Pedestrians website[1]. The results are ordered by miss rate (the lower the better) at 1 false positive per image on average (1 FPPI). The value of 1 FPPI is meaningful for pedestrian detection because in real world applications, it is desirable to limit the number of false alarms.

It can be seen from figure 4 that unsupervised initialization significantly improves the performance ($14.8\% vs 11.5\%$). The number of labeled images in INRIA dataset is relatively small, which limits the capability of supervised learning algorithms. However, an unsupervised method can model large variations in pedestrian pose, scale and clutter with much better success.

Top performing methods [26], [27], [28], [24] also contain several components that our simplistic model does not contain. Probably, the most important of all is color information, whereas we have trained our systems only on gray-scale images. Another important aspect is training on multi-resolution inputs [26], [27], [28]. Currently, we train our systems on fixed scale inputs with very small variation. Additionally, we have used much lower resolution images than top performing systems to train our models ($78 \times 38$ vs $128 \times 64$ in [24]). Finally, some models [28] use deformable body parts models to improve their performance, whereas we rely on a much simpler pipeline of feature extraction and linear classification.

Our aim in this work was to show that an adaptable feature extraction system that learns its parameters from available data can perform comparably to best systems for pedestrian detection. We believe by including color features and using multi-resolution input our system's performance would increase.

## 4   Summary and Future Work

In this work we have presented a method for learning hierarchical feature extractors. Two different methods were presented for convolutional sparse coding, it was shown that convolutional training of feature extractors reduces the redundancy among filters compared with those obtained from patch based models. Additionally, we have introduced two different convolutional encoder functions for performing efficient feature extraction which is crucial for using sparse coding in real world applications. We have applied the proposed sparse modeling systems using a successful multi-stage architecture on object recognition and pedestrian detection problems and performed comparably to similar systems.

In the pedestrian detection task, we have presented the advantage of using unsupervised learning for feature extraction. We believe unsupervised learning significantly helps to properly model extensive variations in the dataset where a pure supervised learning algorithm fails. We aim to further improve our system by better modeling the input by including color and multi-resolution information.

## Footnotes

*Laboratoire d'Informatique de l'Ecole Normale Supérieure (INRIA/ENS/CNRS UMR 8548)

[1]http://www.vision.caltech.edu/Image_Datasets/CaltechPedestrians/files/data-INRIA

# References

[1] LeCun, Y, Bottou, L, Bengio, Y, and Haffner, P. Gradient-based learning applied to document recognition. *Proceedings of the IEEE*, 86(11):2278–2324, November 1998.

[2] Serre, T, Wolf, L, and Poggio, T. Object recognition with features inspired by visual cortex. In *CVPR'05 - Volume 2*, pages 994–1000, Washington, DC, USA, 2005. IEEE Computer Society.

[3] Lee, H, Grosse, R, Ranganath, R, and Ng, A. Convolutional deep belief networks for scalable unsupervised learning of hierarchical representations. In *ICML'09*, pages 609–616. ACM, 2009.

[4] Ranzato, M, Poultney, C, Chopra, S, and LeCun, Y. Efficient learning of sparse representations with an energy-based model. In *NIPS'07*. MIT Press, 2007.

[5] Kavukcuoglu, K, Ranzato, M, Fergus, R, and LeCun, Y. Learning invariant features through topographic filter maps. In *CVPR'09*. IEEE, 2009.

[6] Zeiler, M, Krishnan, D, Taylor, G, and Fergus, R. Deconvolutional Networks. In *CVPR'10*. IEEE, 2010.

[7] Aharon, M, Elad, M, and Bruckstein, A. M. K-SVD and its non-negative variant for dictionary design. In Papadakis, M, Laine, A. F, and Unser, M. A, editors, *Society of Photo-Optical Instrumentation Engineers (SPIE) Conference Series*, volume 5914, pages 327–339, August 2005.

[8] Mairal, J, Bach, F, Ponce, J, and Sapiro, G. Online dictionary learning for sparse coding. In *ICML'09*, pages 689–696. ACM, 2009.

[9] Li, Y and Osher, S. Coordinate Descent Optimization for l1 Minimization with Application to Compressed Sensing; a Greedy Algorithm. *CAM Report*, pages 09–17.

[10] Olshausen, B. A and Field, D. J. Sparse coding with an overcomplete basis set: a strategy employed by v1? *Vision Research*, 37(23):3311–3325, 1997.

[11] Beck, A and Teboulle, M. A fast iterative shrinkage-thresholding algorithm for linear inverse problems. *SIAM J. Img. Sci.*, 2(1):183–202, 2009.

[12] Mallat, S and Zhang, Z. Matching pursuits with time-frequency dictionaries. *IEEE Transactions on Signal Processing*, 41(12):3397:3415, 1993.

[13] Martin, D, Fowlkes, C, Tal, D, and Malik, J. A database of human segmented natural images and its application to evaluating segmentation algorithms and measuring ecological statistics. In *ICCV'01*, volume 2, pages 416–423, July 2001.

[14] Jarrett, K, Kavukcuoglu, K, Ranzato, M, and LeCun, Y. What is the best multi-stage architecture for object recognition? In *ICCV'09*. IEEE, 2009.

[15] Gregor, K and LeCun, Y. Learning fast approximations of sparse coding. In *Proc. International Conference on Machine learning (ICML'10)*, 2010.

[16] LeCun, Y, Bottou, L, Orr, G, and Muller, K. Efficient backprop. In Orr, G and K., M, editors, *Neural Networks: Tricks of the trade*. Springer, 1998.

[17] Schwartz, O and Simoncelli, E. P. Natural signal statistics and sensory gain control. *Nature Neuroscience*, 4(8):819–825, August 2001.

[18] Lyu, S and Simoncelli, E. P. Nonlinear image representation using divisive normalization. In *CVPR'08*. IEEE Computer Society, Jun 23-28 2008.

[19] Fei-Fei, L, Fergus, R, and Perona, P. Learning generative visual models from few training examples: an incremental Bayesian approach tested on 101 object categories. In *Workshop on Generative-Model Based Vision*, 2004.

[20] Pinto, N, Cox, D. D, and DiCarlo, J. J. Why is real-world visual object recognition hard? *PLoS Comput Biol*, 4(1):e27, 01 2008.

[21] Lazebnik, S, Schmid, C, and Ponce, J. Beyond bags of features: Spatial pyramid matching for recognizing natural scene categories. *CVPR'06*, 2:2169–2178, 2006.

[22] Boureau, Y, Bach, F, LeCun, Y, and Ponce, J. Learning mid-level features for recognition. In *CVPR'10*. IEEE, 2010.

[23] Dalal, N and Triggs, B. Histograms of oriented gradients for human detection. In Schmid, C, Soatto, S, and Tomasi, C, editors, *CVPR'05*, volume 2, pages 886–893, June 2005.

[24] Walk, S, Majer, N, Schindler, K, and Schiele, B. New features and insights for pedestrian detection. In *CVPR 2010, San Francisco, California.*

[25] Dollár, P, Wojek, C, Schiele, B, and Perona, P. Pedestrian detection: A benchmark. In *CVPR'09*. IEEE, June 2009.

[26] Dollár, P, Tu, Z, Perona, P, and Belongie, S. Integral channel features. In *BMVC 2009, London, England.*

[27] Dollár, P, Belongie, S, and Perona, P. The fastest pedestrian detector in the west. In *BMVC 2010, Aberystwyth, UK.*

[28] Felzenszwalb, P, Girshick, R, McAllester, D, and Ramanan, D. Object detection with discriminatively trained part based models. In *PAMI 2010.*

